# Entropic Graph Regularization in Non-Parametric Semi-Supervised Classification

**Amarnag Subramanya & Jeff Bilmes**
Department of Electrical Engineering, University of Washington, Seattle.
{asubram,bilmes}@ee.washington.edu

## Abstract

We prove certain theoretical properties of a graph-regularized transductive learning objective that is based on minimizing a Kullback-Leibler divergence based loss. These include showing that the iterative alternating minimization procedure used to minimize the objective converges to the correct solution and deriving a test for convergence. We also propose a graph node ordering algorithm that is cache cognizant and leads to a linear speedup in parallel computations. This ensures that the algorithm scales to large data sets. By making use of empirical evaluation on the TIMIT and Switchboard I corpora, we show this approach is able to outperform other state-of-the-art SSL approaches. In one instance, we solve a problem on a 120 million node graph.

## 1 Introduction

The process of training classifiers with small amounts of labeled data and relatively large amounts of unlabeled data is known as *semi-supervised learning* (SSL). In many applications, such as speech recognition, annotating training data is time-consuming, tedious and error-prone. SSL lends itself as a useful technique in such situations as one only needs to annotate small amounts of data for training models. For a survey of SSL algorithms, see [1, 2]. In this paper we focus on graph-based SSL [1]. Here one assumes that the labeled and unlabeled samples are embedded within a low-dimensional manifold expressed by a graph — each data sample is represented by a vertex within a weighted graph with the weights providing a measure of similarity between vertices. Some graph-based SSL approaches perform random walks on the graph for inference [3, 4] while others optimize a loss function based on smoothness constraints derived from the graph [5, 6, 7, 8]. Graph-based SSL algorithms are inherently non-parametric, transductive and discriminative [2]. The results of the benchmark SSL evaluations in chapter 21 of [1] show that graph-based algorithms are in general better than other SSL algorithms.

Most of the current graph-based SSL algorithms have a number of shortcomings – (a) in many cases, such as [6, 9], a two class problem is assumed; this necessitates the use of sub-optimal extensions like one vs. rest to solve multi-class problems, (b) most graph-based SSL algorithms (exceptions include [7, 8]) attempt to minimize squared error which is not optimal for classification problems [10], and (c) there is a lack of principled approaches to integrating class prior information into graph-based SSL algorithms. Approaches such as *class mass normalization* and *label bidding* are used as a post-processing step rather than being tightly integrated with the inference. To address some of the above issues, we proposed a new graph-based SSL algorithm based on minimizing a Kullback-Leibler divergence (KLD) based loss in [11]. Some of the advantages of this approach include, straightforward extension to multi-class problems, ability to handle label uncertainty and integrate priors. We also showed that this objective can be minimized using alternating minimization (AM), and can outperform other state-of-the-art SSL algorithms for document classification.

Another criticism of previous work in graph-based SSL (and SSL in general) is the lack of algorithms that scale to very large data sets. SSL is based on the premise that unlabeled data is easily

obtained, and adding large quantities of unlabeled data leads to improved performance. Thus practical scalability (e.g., parallelization), is very important in SSL algorithms. [12, 13] discuss the application of TSVMs to large-scale problems. [14] suggests an algorithm for improving the induction speed in the case of graph-based algorithms. [15] solves a graph transduction problem with 650,000 samples. To the best of our knowledge, the largest graph-based problem solved to date had about 900,000 samples (includes both labeled and unlabeled data) [16]. Clearly, this is a fraction of the amount of unlabeled data at our disposal. For example, on the Internet alone we create 1.6 billion blog posts, 60 billion emails, 2 million photos and 200,000 videos every day [17].

The goal of this paper is to provide theoretical analysis of our algorithm proposed in [11] and also show how it can be scaled to very large problems. We first prove that AM on our KLD based objective converges to the true optimum. We also provide a test for convergence and discuss some theoretical connections between the two SSL objectives proposed in [11]. In addition, we propose a graph node ordering algorithm that is cache cognizant and makes obtaining a linear speedup with a parallel implementation more likely. As a result, the algorithms are able to scale to very large datasets. The node ordering algorithm is quite general and can be applied to graph-based SSL algorithm such as [5, 11]. In one instance, we solve a SSL problem over a graph with **120 million** vertices. We use the phone classification problem to demonstrate the scalability of the algorithm. We believe that speech recognition is an ideal application for SSL and in particular graph-based SSL for several reasons: (a) human speech is produced by a small number of articulators and thus amenable to representation in a low-dimensional manifold [18]; (b) annotating speech data is time-consuming and costly; and (c) the data sets tend to be very large.

## 2 Graph-based SSL

Let $\mathcal{D}_l = \{(\mathbf{x}_i, r_i)\}_{i=1}^l$ be the set of labeled samples, $\mathcal{D}_u = \{\mathbf{x}_i\}_{i=l+1}^{l+u}$, the set of unlabeled samples and $\mathcal{D} \triangleq \{\mathcal{D}_l, \mathcal{D}_u\}$. Here $r_i$ is an encoding of the labeled data and will be explained shortly. We are interested in solving the transductive learning problem, i.e., given $\mathcal{D}$, the task is to predict the labels of the samples in $\mathcal{D}_u$. The first step in most graph-based SSL algorithms is the construction of an undirected weighted graph $\mathcal{G} = (V, E)$, where the vertices (nodes) $V = \{1, \ldots, m\}$, $m = l + u$, are the data points in $\mathcal{D}$ and the edges $E \subseteq V \times V$. Let $V_l$ and $V_u$ be the set of labeled and unlabeled vertices respectively. $\mathcal{G}$ may be represented via a symmetric matrix $\mathbf{W}$ which is referred to as the weight or affinity matrix. There are many ways of constructing the graph (see section 6.2 in [2]). In this paper, we use symmetric $k$-nearest neighbor (NN) graphs — that is, we first form $w_{ij} \triangleq [\mathbf{W}]_{ij} = \text{sim}(\mathbf{x}_i, \mathbf{x}_j)$ and then make this graph sparse by setting $w_{ij} = 0$ unless $i$ is one of $j$'s $k$ nearest neighbors or $j$ is one of $i$'s $k$ nearest neighbors. It is assumed that $\text{sim}(x, y) = \text{sim}(y, x)$. Let $\mathcal{N}(i)$ be the set of neighbors of vertex $i$. Choosing the correct similarity measure and $|\mathcal{N}(i)|$ are crucial steps in the success of any graph-based SSL algorithm as it determines the graph [2].

For each $i \in V$ and $j \in V_l$, define probability measures $p_i$ and $r_j$ respectively over the measurable space $(Y, \mathcal{Y})$. Here $\mathcal{Y}$ is the $\sigma$-field of measurable subsets of $Y$ and $Y \subset \mathbb{N}$ (the set of natural numbers) is the space of classifier outputs. Thus $|Y| = 2$ yields binary classification while $|Y| > 2$ implies multi-class. As we only consider classification problems here, $p_i$ and $r_i$ are multinomial distributions, $p_i(y)$ is the probability that $\mathbf{x}_i$ belongs to class $y$ and the classification result is given by $\text{argmax}_y\, p_i(y)$. $\{r_j\}$, $j \in V_l$ encodes the labels of the supervised portion of the training data. If the labels are known with certainty, then $r_j$ is a "one-hot" vector (with the single 1 at the appropriate position in the vector). $r_j$ is also capable of representing cases where the label is uncertain, i.e., for example when the labels are in the form of a distribution (possibly derived from normalizing scores representing confidence). It is important to distinguish between the classical multi-label problem and the use of uncertainty in $r_j$. If it is the case that $r_j(\bar{y}_1), r_j(\bar{y}_2) > 0$, $\bar{y}_1 \neq \bar{y}_2$, it does not imply that the input $\mathbf{x}_j$ possesses two output labels $\bar{y}_1$ and $\bar{y}_2$. Rather, $r_j$ represents our belief in the various values of the output. As $p_i$, $r_i$ are probability measures, they lie within a $|Y|$-dimensional probability simplex which we represent using $\triangle_{|Y|}$ and so $p_i$, $r_i \in \triangle_{|Y|}$ (henceforth denoted as $\triangle$). Also let $\text{p} \triangleq (p_1, \ldots, p_m) \in \triangle^m \triangleq \triangle \times \ldots \times \triangle$ ($m$ times) and $\text{r} \triangleq (r_1, \ldots, r_l) \in \triangle^l$.

Consider the optimization problem proposed in [11] where $\text{p}^* = \min_{\text{p} \in \triangle^m} \mathcal{C}_1(\text{p})$ and

$$\mathcal{C}_1(\text{p}) = \sum_{i=1}^l D_{KL}(r_i || p_i) + \mu \sum_{i=1}^m \sum_{j \in \mathcal{N}(i)} w_{ij} D_{KL}(p_i || p_j) - \nu \sum_{i=1}^m H(p_i).$$

Here $H(p) = -\sum_{\mathrm{y}} p(\mathrm{y}) \log p(\mathrm{y})$ is the Shannon entropy of $p$ and $D_{KL}(p||q)$ is the KLD between measures $p$ and $q$ and is given by $D_{KL}(p||q) = \sum_{\mathrm{y}} p(\mathrm{y}) \log \frac{p(\mathrm{y})}{q(\mathrm{y})}$. If $\mu, \nu, w_{ij} \geq 0, \forall i, j$ then $\mathcal{C}_1(\mathrm{p})$ is convex [19]. $(\mu, \nu)$ are hyper-parameters whose choice we discuss in Section 5. The first term in $\mathcal{C}_1$ penalizes the solution $p_i$ $i \in \{1, \ldots, l\}$, when it is far away from the labeled training data $\mathcal{D}_l$, but it does not insist that $p_i = r_i$, as allowing for deviations from $r_i$ can help especially with noisy labels [20] or when the graph is extremely dense in certain regions. The second term of $\mathcal{C}_1$ penalizes a lack of consistency with the geometry of the data, i.e., a graph regularizer. If $w_{ij}$ is large, we prefer a solution in which $p_i$ and $p_j$ are close in the KLD sense. The last term encourages each $p_i$ to be close to the uniform distribution if not preferred to the contrary by the first two terms. This acts as a guard against degenerate solutions commonly encountered in graph-based SSL [6], e.g., in cases where a sub-graph is not connected to any labeled vertex. We conjecture that by maximizing the entropy of each $p_i$, the classifier has a better chance of producing high entropy results in graph regions of low confidence (e.g. close to the decision boundary and/or low density regions). To recap, $\mathcal{C}_1$ makes use of the manifold assumption, is naturally multi-class and able to encode label uncertainty.

As $\mathcal{C}_1$ is convex in p with linear constraints, we have a *convex programming problem*. However, a closed form solution does not exist and so standard numerical optimization approaches such as interior point methods (IPM) or method of multipliers (MOM) can be used to solve the problem. But, each of these approaches have their own shortcomings and are rather cumbersome to implement (e.g. an implementation of MOM to solve this problem would have 7 extraneous parameters). Thus, in [11], we proposed the use of AM for minimizing $\mathcal{C}_1$. We will address the question of whether AM is superior to IPMs or MOMs for minimizing $\mathcal{C}_1$ shortly.

Consider a problem of minimizing $d(p, q)$ over $p \in \mathcal{P}, q \in \mathcal{Q}$. Sometimes solving this problem directly is hard and in such cases AM lends itself as a valuable tool for efficient optimization. It is an iterative process in which $p^{(n)} = \mathrm{argmin}_{p \in \mathcal{P}} d(p, q^{(n-1)})$ and $q^{(n+1)} = \mathrm{argmin}_{q \in \mathcal{Q}} d(p^{(n)}, q)$. The Expectation-Maximization (EM) [21] algorithm is an example of AM. $\mathcal{C}_1$ is not amenable to optimization using AM and so we have proposed a modified version of the objective where $(\mathrm{p}^*, \mathrm{q}^*) = \min_{\mathrm{p}, \mathrm{q} \in \Delta^m} \mathcal{C}_2(\mathrm{p}, \mathrm{q})$ and

$$\mathcal{C}_2(\mathrm{p}, \mathrm{q}) = \sum_{i=1}^{l} D_{KL}(r_i || q_i) + \mu \sum_{i=1}^{m} \sum_{j \in \mathcal{N}'(i)} w'_{ij} D_{KL}(p_i || q_j) - \nu \sum_{i=1}^{m} H(p_i).$$

In the above, a third measure $q_i$, $\forall i \in V$ is defined over the measurable space $(\mathrm{Y}, \mathcal{Y})$, $\mathbf{W}' = \mathbf{W} + \alpha \mathbf{I}_n$, $\mathcal{N}'(i) = \{\{i\} \cup \mathcal{N}(i)\}$ and $\alpha \geq 0$. Here the $q_i$'s play a similar role as the $p_i$'s and can potentially be used to obtain a final classification result ($\mathrm{argmax}_y q_i(y)$), but $\alpha$, which is a hyper-parameter, plays an important role in ensuring that $p_i$ and $q_i$ are close $\forall i$. It should be at least intuitively clear that as $\alpha$ gets large, the reformulated objective ($\mathcal{C}_2$) apparently approaches the original objective ($\mathcal{C}_1$). Our results from [11] suggest that setting $\alpha = 2$ ensures that $\mathrm{p}^* = \mathrm{q}^*$ (more on this in the next section). It is important to highlight that $\mathcal{C}_2(\mathrm{p}, \mathrm{q})$ is itself still a valid SSL criterion. While the first term encourages $q_i$ for the labeled vertices to be close to the labels, $r_i$, the last term encourages higher entropy $p_i$'s. The second term, in addition to acting as a graph regularizer, also acts as glue between the $p$'s and $q$'s. The update equations for solving $\mathcal{C}_2(\mathrm{p}, \mathrm{q})$ are given by

$$p_i^{(n)}(y) = \frac{\exp\{\frac{\mu}{\gamma_i} \sum_j w'_{ij} \log q_j^{(n-1)}(y)\}}{\sum_y \exp\{\frac{\mu}{\gamma_i} \sum_j w'_{ij} \log q_j^{(n-1)}(y)\}} \quad \text{and} \quad q_i^{(n)}(y) = \frac{r_i(y)\delta(i \leq l) + \mu \sum_j w'_{ji} p_j^{(n)}(y)}{\delta(i \leq l) + \mu \sum_j w'_{ji}}$$

where $\gamma_i = \nu + \mu \sum_j w'_{ij}$. Intuitively, discrete probability measures are being propagated between vertices along edges, so we refer to this algorithm as *measure propagation* (MP).

When AM is used to solve an optimization problem, a closed form solution to each of the steps of the AM is desired but not always guaranteed [7]. It can be seen that solving $\mathcal{C}_2$ using AM has a *single* additional hyper-parameter while other approaches such as MOM can have as many as 7. Further, as we show in section 4, the AM update equations can be easily parallelized.

We briefly comment on the relationship to previous work. As noted in section 1, a majority of the previous graph-based SSL algorithms are based on minimizing squared error [6, 5]. While

these objectives are convex and in some cases admit closed-form (i.e., non-iterative) solutions, they require inverting a matrix of size $m \times m$. Thus in the case of very large data sets (e.g., like the one we consider in section 5), it might not be feasible to use this approach. Therefore, an iterative update is employed in practice. Also, squared-error is only optimal under a Gaussian loss model and thus more suitable for regression rather than classification problems. Squared-loss penalizes *absolute error*, while KLD, on the other-hand penalizes *relative error* (pages 226 and 235 of [10]). Henceforth, we refer to a multi-class extension of algorithm 11.2 in [20] as *SQ-Loss*.

The Information Regularization (IR) [7] approach and subsequently the algorithm of Tsuda [8] use KLD based objectives and utilize AM to solve the problem. However these algorithms are motivated from a different perspective. In fact, as stated above, one of the steps of the AM procedure in the case of IR does not admit a closed form solution. In addition, neither IR nor the work of Tsuda use an entropy regularizer, which, as our results will show, leads to improved performance. While the two steps of the AM procedure in the case of Tsuda's work have closed form solutions and the approach is applicable to hyper-graphs, one of the updates (equation 13 in [8]) is a special of the update for $p_i^{(n)}$. For more connections to previous approaches, see Section 4 in [11].

## 3    Optimal Convergence of AM on $\mathcal{C}_2$

We show that AM on $\mathcal{C}_2$ converges to the minimum of $\mathcal{C}_2$, and that there exists a finite $\alpha$ such that the optimal solutions of $\mathcal{C}_1$ and $\mathcal{C}_2$ are identical. Therefore, $\mathcal{C}_2$ is a perfect tractable surrogate for $\mathcal{C}_1$.

In general, AM is not always guaranteed to converge to the correct solution. For example, consider minimizing $f(x,y) = x^2 + 3xy + y^2$ over $x, y \in \mathbb{R}$ where $f(x,y)$ is unbounded below (consider $y = -x$). But AM says that $(x^*, y^*) = (0,0)$ which is incorrect (see [22] for more examples). For AM to converge to the correct solution certain conditions must be satisfied. These might include topological properties of the optimization problem [23, 24] or certain geometrical properties [25]. The latter is referred to as the *Information Geometry* approach where the *5-points property* (5-pp) [25] plays an important role in determining convergence and is the method of choice here.

**Theorem 3.1 (Convergence of AM on $\mathcal{C}_2$).** *If*

$$\mathrm{p}^{(n)} = \operatorname*{argmin}_{\mathrm{p} \in \triangle^m} \mathcal{C}_2(\mathrm{p}, \mathrm{q}^{(n-1)}), \quad \mathrm{q}^{(n)} = \operatorname*{argmin}_{\mathrm{q} \in \triangle^m} \mathcal{C}_2(\mathrm{p}^{(n)}, \mathrm{q}) \text{ and } q_i^{(0)}(y) > 0 \ \forall \ y \in \mathrm{Y}, \ \forall i \text{ then}$$

*(a)* $\mathcal{C}_2(\mathrm{p}, \mathrm{q}) + \mathcal{C}_2(\mathrm{p}, \mathrm{p}^{(0)}) \geq \mathcal{C}_2(\mathrm{p}, \mathrm{q}^{(1)}) + \mathcal{C}_2(\mathrm{p}^{(1)}, \mathrm{q}^{(1)})$ *for all* $\mathrm{p}, \mathrm{q} \in \triangle^m$, *and*

*(b)* $\lim_{n \to \infty} \mathcal{C}_2(\mathrm{p}^{(n)}, \mathrm{q}^{(n)}) = \inf_{\mathrm{p}, \mathrm{q} \in \triangle^m} \mathcal{C}_2(\mathrm{p}, \mathrm{q})$.

*Proof Sketch*: (a) is the 5-pp for $\mathcal{C}_2(\mathrm{p}, \mathrm{q})$. 5-pp holds if the 3-points (3-pp) and 4-points (4-pp) properties hold. In order to show 3-pp, let $f(t) \triangleq \mathcal{C}_2(\mathrm{p}^{(t)}, \mathrm{q}^{(0)})$ where $\mathrm{p}^{(t)} = (1-t)\mathrm{p} + t\mathrm{p}^{(1)}$, $0 < t \leq 1$. Next we use the fact that the first-order Taylor's approximation underestimates a convex function to upper bound the gradient of $f(t)$ w.r.t $t$. We then pass this to the limit as $t \to 1$ and use the monotone convergence theorem to exchange the limit and the summation. This gives the 3-pp. The proof for 4-pp follows in a similar manner. (b) follows as a result of Theorem 3 in [25]. $\square$

**Theorem 3.2 (Test for Convergence).** *If* $\{(\mathrm{p}^{(n)}, \mathrm{q}^{(n)})\}_{n=1}^{\infty}$ *is generated by AM of* $\mathcal{C}_2(\mathrm{p}, \mathrm{q})$ *and* $\mathcal{C}_2(\mathrm{p}^*, \mathrm{q}^*) \triangleq \inf_{\mathrm{p}, \mathrm{q} \in \triangle^m} \mathcal{C}_2(\mathrm{p}, \mathrm{q})$ *then*

$$\mathcal{C}_2(\mathrm{p}^{(n)}, \mathrm{q}^{(n)}) - \mathcal{C}_2(\mathrm{p}^*, \mathrm{q}^*) \leq \sum_{i=1}^{m} \big(\delta(i \leq l) + d_i\big)\beta_i \text{ where } \beta_i \triangleq \log \sup_y \frac{q_i^{(n)}(y)}{q_i^{(n-1)}(y)}, d_j = \sum_i w'_{ij}.$$

*Proof Sketch*: As the 5-pp holds for all $\mathrm{p}, \mathrm{q} \in \triangle^m$, it also holds for $\mathrm{p} = \mathrm{p}^*$ and $\mathrm{q} = \mathrm{q}^*$. We use fact that $E(f(Z)) \leq \sup_z f(z)$ where $Z$ is a random variable and $f(\cdot)$ is an arbitrary function. $\square$

The above means that AM on $\mathcal{C}_2$ converges to its optimal value. We also have the following theorems that show the existence of a finite lower-bound on $\alpha$ such that the optimum of $\mathcal{C}_1$ and $\mathcal{C}_2$ are the same.

**Lemma 3.3.** *If* $\mathcal{C}_2(\mathrm{p}, \mathrm{q}; w'_{ii} = 0)$ *is* $\mathcal{C}_2$ *when the diagonal elements of the affinity matrix are all zero then we have that*

$$\min_{\mathrm{p}, \mathrm{q} \in \triangle^m} \mathcal{C}_2(\mathrm{p}, \mathrm{q}; w'_{ii} = 0) \leq \min_{\mathrm{p} \in \triangle^m} \mathcal{C}_1(\mathrm{p})$$

**Theorem 3.4.** *Given any $A, B, S \in \triangle^m$ (i.e., $A = [a_1, \ldots, a_n]$, $B = [b_1, \ldots, b_n]$, $S = [s_1, \ldots, s_n]$) such that $a_i(y), b_i(y), s_i(y) > 0$, $\forall\, i, y$ and $A \neq B$ (i.e., not all $a_i(y) = b_i(y)$) then there exists a finite $\alpha$ such that*

$$\mathcal{C}_2(A, B) \geq \mathcal{C}_2(S, S) = \mathcal{C}_1(S)$$

**Theorem 3.5** (**Equality of Solutions of $\mathcal{C}_1$ and $\mathcal{C}_2$**)**.** *Let* $\hat{p} = \underset{p \in \triangle^m}{\operatorname{argmin}} C_1(p)$ *and* $(p_{\tilde{\alpha}}^*, q_{\tilde{\alpha}}^*) = \underset{p, q \in \triangle^m}{\operatorname{argmin}} C_2(p, q; \tilde{\alpha})$ *for an arbitrary* $\alpha = \tilde{\alpha} > 0$ *where* $p_{\tilde{\alpha}}^* = (p_{1;\tilde{\alpha}}^*, \cdots, p_{m;\tilde{\alpha}}^*)$ *and* $q_{\tilde{\alpha}}^* = (q_{1;\tilde{\alpha}}^*, \cdots, q_{m;\tilde{\alpha}}^*)$. *Then there exists a finite $\hat{\alpha}$ such that at convergence of AM, we have that* $\hat{p} = p_{\hat{\alpha}}^* = q_{\hat{\alpha}}^*$. *Further, it is the case that if* $p_{\hat{\alpha}}^* \neq q_{\hat{\alpha}}^*$, *then*

$$\hat{\alpha} \geq \frac{C_1(\hat{p}) - C_2(p^*, q^*; \alpha = 0)}{\mu \sum_{i=1}^n D_{KL}(p_{i;\tilde{\alpha}}^* \| q_{i;\tilde{\alpha}}^*)}.$$

*And if* $p_{\hat{\alpha}}^* = q_{\hat{\alpha}}^*$ *then* $\hat{\alpha} \geq \tilde{\alpha}$.

## 4 Parallelism and Scalability to Large Datasets

One big advantage of AM on $\mathcal{C}_2$ over optimizing $\mathcal{C}_1$ directly is that it is naturally amenable to a parallel implementation, and is also amenable to further optimizations (see below) that yield a near linear speedup. Consider the update equations of Section 2. We see that one set of measures is held fixed while the other set is updated without any required communication amongst set members, so there is no write contention. This immediately yields a $T \geq 1$-threaded implementation where the graph is evenly $T$-partitioned and each thread operates over only a size $m/T = (l + u)/T$ subset of the graph nodes.

We constructed a 10-NN graph using the standard TIMIT training and development sets (see section 5). The graph had 1.4 million vertices. We ran a timing test on a 16 core symmetric multiprocessor with 128GB of RAM, each core operating at 1.6GHz. We varied the number $T$ of threads from 1 (single-threaded) up to 16, in each case running 3 iterations of AM (i.e., 3 each of p and q updates). Each experiment was repeated 10 times, and we measured the minimum CPU time over these 10 runs (total CPU time only was taken into account). The speedup for $T$ threads is typically defined as the ratio of time taken for single thread to time taken for $T$ threads. The solid (black) line in figure 1(a) represents the ideal case (a linear speedup), i.e., when using $T$ threads results in a speedup of $T$. The pointed (green) line shows the actual speedup of the above procedure, typically less than ideal due to inter-process communication and poor shared L1 and/or L2 microprocessor cache interaction. When $T \leq 4$, the speedup (green) is close to ideal, but for increasing $T$ the performance diminishes away from the ideal case.

Our contention is that the sub-linear speedup is due to the poor cache cognizance of the algorithm. At a given point in time, suppose thread $t \in \{1, \ldots, T\}$ is operating on node $i_t$. The collective set of neighbors that are being used by these $T$ threads are $\{\cup_{t=1}^T \mathcal{N}(i_t)\}$ and this, along with nodes $\cup_{t=1}^T \{i_t\}$ (and all memory for the associated measures), constitute the current *working set*. The working set should be made as small as possible to increase the chance it will fit in the microprocessor caches, but this becomes decreasingly likely as $T$ increases since the working set is monotonically increasing with $T$. Our goal, therefore, is for the nodes that are being simultaneously operated on to have a large amount of neighbor overlap thus minimizing the working set size. Viewed as an optimization problem, we must find a partition $(V_1, V_2, \ldots, V_{m/T})$ of $V$ that minimizes $\max_{j \in \{1, \ldots, m/T\}} |\cup_{v \in V_j} \mathcal{N}(v)|$. With such a partition, we may also order the subsets so that the neighbors of $V_i$ would have maximal overlap with the neighbors of $V_{i+1}$. We then schedule the $T$ nodes in $V_j$ to run simultaneously, and schedule the $V_j$ sets successively.

Of course, the time to produce such a partition cannot dominate the time to run the algorithm itself. Therefore, we propose a simple fast node ordering procedure (Algorithm 1) that can be run once before the parallelization begins. The algorithm orders the nodes such that successive nodes are likely to have a high amount of neighbor overlap with each other and, by transitivity, with nearby nodes in the ordering. It does this by, given a node $v$, choosing another node $v'$ from amongst $v$'s neighbors' neighbors (meaning the neighbors of $v$'s neighbors) that has the highest neighbor overlap. We need not search all $V$ nodes for this, since anything other than $v$'s neighbors' neighbors

**Algorithm 1** Graph Ordering Algorithm
___
Select an arbitrary node $v$.
**while** there are any unselected nodes remaining **do**
    Let $\mathcal{N}(v)$ be the set of neighbors, and $\mathcal{N}^2(v)$ be the set of neighbors' neighbors, of $v$.
    Select a currently unselected $v' \in \mathcal{N}^2(v)$ such that $|\mathcal{N}(v) \cap \mathcal{N}(v')|$ is maximized. If the intersection is empty, select an arbitrary unselected $v'$.
    $v \leftarrow v'$.
**end while**
___

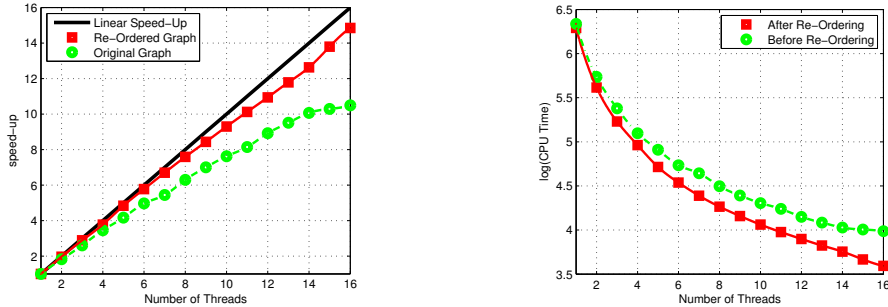

Figure 1: (a) speedup vs. number of threads for the TIMIT graph (see section 5). The process was run on a 128GB, 16 core machine with each core at 1.6GHz. (b) The *actual* CPU times in seconds on a *log scale* vs. number of threads for with and without ordering cases.

has no overlap with the neighbors of $v$. Given such an ordering, the $t^{th}$ thread operates on nodes $\{t, t + m/T, t + 2m/T, \dots\}$. If the threads proceed synchronously (which we do not enforce) the set of nodes being processed at any time instant are $\{1 + jm/T, 2 + jm/T, \dots, T + jm/T\}$. This assignment is beneficial not only for maximizing the set of neighbors being simultaneously used, but also for successive chunks of $T$ nodes since once a chunk of $T$ nodes have been processed, it is likely that many of the neighbors of the next chunk of $T$ nodes will already have been pre-fetched into the caches. With the graph represented as an adjacency list, and sets of neighbor indices sorted, our algorithm is $O(mk^3)$ in time and linear in memory since the intersection between two sorted lists may be computed in $O(k)$ time. This is sometimes better than $O(m \log m)$ for cases where $k^3 < \log m$, true for very large $m$.

We ordered the TIMIT graph nodes, and ran timing tests as explained above. To be fair, the time required for node ordering **is** charged against every run. The results are shown in figure 1(a) (pointed red line) where the results are much closer to ideal, and there are no obvious diminishing returns like in the unordered case. Running times are given in figure 1(b). Moreover, the ordered case showed better performance even for a single thread $T = 1$ (CPU time of 539s vs. 565s for ordered vs. unordered respectively, on 3 iterations of AM).

We conclude this section by noting that (a) re-ordering may be considered a pre-processing (offline) step, (b) the SQ-Loss algorithm may also be implemented in a multi-threaded manner and this is supported by our implementation, (c) our re-ordering algorithm is general and fast and can be used for any graph-based algorithm where the iterative updates for a given node are a function of its neighbors (i.e., the updates are harmonic w.r.t. the graph [5]), and (d) while the focus here was on parallelization across different processors on a symmetric multiprocessor, this would also apply for distributed processing across a network with a shared network disk.

## 5 Results

In this section we present results on two popular phone classification tasks. We use SQ-Loss as the competing graph-based algorithm and compare its performance against that of MP because (a) SQ-Loss has been shown to outperform its other variants, such as, label propagation [4] and the harmonic function algorithm [5], (b) SQ-Loss scales easily to very large data sets unlike approaches like spectral graph transduction [6], and (c) SQ-Loss gives similar performance as other algorithms that minimize squared error such as manifold regularization [20].

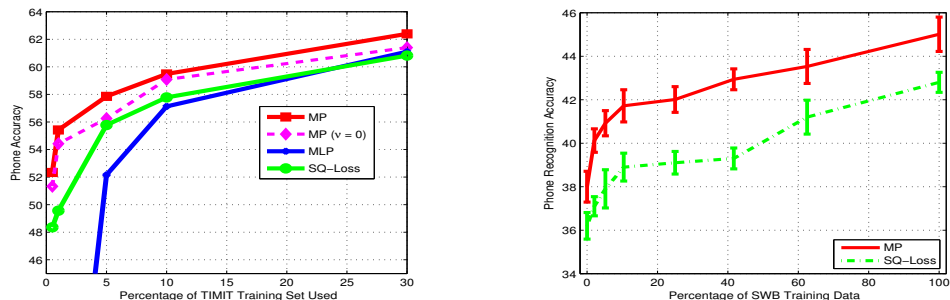

Figure 2: Phone accuracy on the TIMIT test set (a,left) and phone accuracy vs. amount of SWB training data (b,right). With all SWB data added, the graph has **120 million** nodes.

*TIMIT Phone Classification:* TIMIT is a corpus of read speech that includes time-aligned phonetic transcriptions. As a result, it has been popular in the speech community for evaluating supervised phone classification algorithms [26]. Here, we use it to evaluate SSL algorithms by using fractions of the standard TIMIT training set, i.e., simulating the case when only small amounts of data are labeled. We constructed a symmetrized 10-NN graph ($\mathcal{G}^{timit}$) over the TIMIT training and development sets (minimum graph degree is 10). The graph had about 1.4 million vertices. We used $\text{sim}(\mathbf{x}_i, \mathbf{x}_j) = \exp\{-(\mathbf{x}_i - \mathbf{x}_j)^T \Sigma^{-1}(\mathbf{x}_i - \mathbf{x}_j)\}$ where $\Sigma$ is the covariance matrix computed over the entire training set. In order to obtain the features, $\mathbf{x}_i$, we first extracted mel-frequency cepstral coefficients (MFCC) along with deltas in the manner described in [27]. As phone classification performance is improved with context information, each $\mathbf{x}_i$ was constructed using a 7 frame context window. We follow the standard practice of building models to classify 48 phones ($|Y| = 48$) and then mapping down to 39 phones for scoring [26].

We compare the performance of MP against MP with no entropy regularization ($\nu = 0$), SQ-Loss, and a supervised state-of-the-art L2 regularized multi-layered perceptron (MLP) [10]. The hyperparameters in each case, i.e., number of hidden units and regularization weight in case of MLP, $\mu$ and $\nu$ in the case of MP and SQ-Loss, were tuned on the development set. For the MP and SQ-Loss, the hyper-parameters were tuned over the following sets $\mu \in \{1\text{e--}8, 1\text{e--}4, 0.01, 0.1\}$ and $\nu \in \{1\text{e--}8, 1\text{e--}6, 1\text{e--}4, 0.01, 0.1\}$. We found that setting $\alpha = 1$ in the case of MP ensured that p = q at convergence. As both MP and SQ-Loss are transductive, in order to measure performance on an independent test set, we induce the labels using the Nadaraya-Watson estimator (see section 6.4 in [2]) with 50 NNs using the similarity measure defined above.

Figure 2(a) shows the phone classification results on the NIST Core test set (independent of the development set). We varied the number of labeled examples by sampling a fraction $f$ of the TIMIT training set. We show results for $f \in \{0.005, 0.05, 0.1, 0.25, 0.3\}$. In all cases, for MP and SQ-Loss, we use the same graph $\mathcal{G}^{timit}$, but the set of labeled vertices changes based on $f$. In all cases the MLP was trained fully-supervised. We only show results on the test set, but the results on the development set showed similar trends. It can be seen that (i) using an entropy regularizer leads to much improved results in MP, (ii) as expected, the MLP being fully-supervised, performs poorly compared to the semi-supervised approaches, and most importantly, (iii) MP significantly outperforms all other approaches. We believe that MP outperforms SQ-Loss as the loss function in the case of MP is better suited for classification. We also found that for larger values of $f$ (e.g., at $f = 1$), the performances of MLP and MP did not differ significantly. But those are more representative of the supervised training scenarios which is not the focus here.

*Switchboard-I Phone Classification:* Switchboard-I (SWB) is a collection of about 2,400 two-sided telephone conversations among 543 speakers [28]. SWB is often used for the training of large vocabulary speech recognizers. The corpus is annotated at the word-level. In addition, less reliable phone level annotations generated in an automatic manner by a speech recognizer with a non-zero error rate are also available [29]. The *Switchboard Transcription Project* (STP) [30] was undertaken to *accurately* annotate SWB at the phonetic and syllable levels. As a result of the arduous and costly nature of this transcription task, only 75 minutes (out of 320 hours) of speech segments selected from different SWB conversations were annotated at the phone level and about 150 minutes annotated at the syllable level. Having such annotations for all of SWB could be useful for speech processing in general, so this is an ideal real-world task for SSL.

We make use of only the phonetic labels ignoring the syllable annotations. Our goal is to phonetically annotate SWB in STP style while treating STP as labeled data, and in the process show that our aforementioned parallelism efforts scale to extremely large datasets. We extracted features $\mathbf{x}_i$ from the conversations by first windowing them using a Hamming window of size 25ms at 100Hz. We then extracted 13 perceptual linear prediction (PLP) coefficients from these windowed features and appended both deltas and double-deltas resulting in a 39 dimensional feature vector. As with TIMIT, we are interested in phone classification and we use a 7 frame context window to generate $\mathbf{x}_i$, stepping successive context windows by 10ms as is standard in speech recognition.

We randomly split the 75 minute phonetically annotated part of STP into three sets, one each for training, development and testing containing 70%, 10% and 20% of the data respectively (the size of the development set is considerably smaller than the size of the training set). This procedure was repeated 10 times (i.e. we generated 10 different training, development and test sets by random sampling). In each case, we trained a phone classifier using the training set, tuned the hyper-parameters on the development set and evaluated the performance on the test set. In the following, we refer to SWB that is not a part of STP as *SWB-STP*. We added the unlabeled SWB-STP data in stages. The percentage, $s$, included, 0%, 2%, 5%, 10%, 25%, 40%, 60%, and 100% of SWB-STP. We ran both MP and SQ-Loss in each case. When $s = 100\%$, there were about 120 million nodes in the graph!

Due to the large size $m = 120M$ of the dataset, it was not possible to generate the graph using the conventional brute-force search which is $O(m^2)$. Nearest neighbor search is a well-researched problem with many approximate solutions [31]. Here we make use of the Approximate Nearest Neighbor (ANN) library (see `http://www.cs.umd.edu/~mount/ANN/`) [32]. It constructs a modified version of a kd-tree which is then used to query the NNs. The query process requires that one specify an error term, $\epsilon$, and guarantees that $(d(\mathbf{x}_i, \mathcal{N}(\mathbf{x}_i))/d(\mathbf{x}_i, \mathcal{N}_\epsilon(\mathbf{x}_i))) \leq 1 + \epsilon$. where $\mathcal{N}(\mathbf{x}_i)$ is a function that returns the actual NN of $\mathbf{x}_i$ while $\mathcal{N}_\epsilon(\mathbf{x}_i)$ returns the approximate NN.

We constructed graphs using the STP data and $s\%$ of (unlabeled) SWB-STP data. For all the experiments here we used a symmetrized 10-NN graph and $\epsilon = 2.0$. The labeled and unlabeled points in the graph changed based on training, development and test sets used. In each case, we ran both the MP and SQ-Loss objectives. For each set, we ran a search over $\mu \in \{1e{-}8, 1e{-}4, 0.01, 0.1\}$ and $\nu \in \{1e{-}8, 1e{-}6, 1e{-}4, 0.01, 0.1\}$ for both the approaches. The best value of the hyper-parameters were chosen based on the performance on the development set and the same value was used to measure the accuracy on the test set. The mean phone accuracy over the different test sets (and the standard deviations) are shown in figure 2(b) for the different values of $s$. It can be seen that MP outperforms SQ-Loss in all cases. Equally importantly, we see that the performance on the STP data improves with the addition of increasing amounts of unlabeled data.

## References

[1] O. Chapelle, B. Scholkopf, and A. Zien, *Semi-Supervised Learning*. MIT Press, 2007.

[2] X. Zhu, "Semi-supervised learning literature survey," tech. rep., Computer Sciences, University of Wisconsin-Madison, 2005.

[3] M. Szummer and T. Jaakkola, "Partially labeled classification with Markov random walks," in *Advances in Neural Information Processing Systems*, vol. 14, 2001.

[4] X. Zhu and Z. Ghahramani, "Learning from labeled and unlabeled data with label propagation," tech. rep., Carnegie Mellon University, 2002.

[5] X. Zhu, Z. Ghahramani, and J. Lafferty, "Semi-supervised learning using gaussian fields and harmonic functions," in *Proc. of the International Conference on Machine Learning (ICML)*, 2003.

[6] T. Joachims, "Transductive learning via spectral graph partitioning," in *Proc. of the International Conference on Machine Learning (ICML)*, 2003.

[7] A. Corduneanu and T. Jaakkola, "On information regularization," in *Uncertainty in Artificial Intelligence*, 2003.

[8] K. Tsuda, "Propagating distributions on a hypergraph by dual information regularization," in *Proceedings of the 22nd International Conference on Machine Learning*, 2005.

[9] M. Belkin, P. Niyogi, and V. Sindhwani, "On manifold regularization," in *Proc. of the Conference on Artificial Intelligence and Statistics (AISTATS)*, 2005.

[10] C. Bishop, ed., *Neural Networks for Pattern Recognition*. Oxford University Press, 1995.

[11] A. Subramanya and J. Bilmes, "Soft-supervised text classification," in *EMNLP*, 2008.

[12] R. Collobert, F. Sinz, J. Weston, L. Bottou, and T. Joachims, "Large scale transductive svms," *Journal of Machine Learning Research*, 2006.

[13] V. Sindhwani and S. S. Keerthi, "Large scale semi-supervised linear svms," in *SIGIR '06: Proceedings of the 29th annual international ACM SIGIR*, 2006.

[14] O. Delalleau, Y. Bengio, and N. L. Roux, "Efficient non-parametric function induction in semi-supervised learning," in *Proc. of the Conference on Artificial Intelligence and Statistics (AISTATS)*, 2005.

[15] M. Karlen, J. Weston, A. Erkan, and R. Collobert, "Large scale manifold transduction," in *International Conference on Machine Learning, ICML*, 2008.

[16] I. W. Tsang and J. T. Kwok, "Large-scale sparsified manifold regularization," in *Advances in Neural Information Processing Systems (NIPS) 19*, 2006.

[17] A. Tomkins, "Keynote speech." CIKM Workshop on Search and Social Media, 2008.

[18] A. Jansen and P. Niyogi, "Semi-supervised learning of speech sounds," in *Interspeech*, 2007.

[19] T. M. Cover and J. A. Thomas, *Elements of Information Theory*. Wiley Series in Telecommunications, New York: Wiley, 1991.

[20] Y. Bengio, O. Delalleau, and N. L. Roux, *Semi-Supervised Learning*, ch. Label Propogation and Quadratic Criterion. MIT Press, 2007.

[21] Dempster, Laird, and Rubin, "Maximum likelihood from incomplete data via the em algorithm," *Journal of the Royal Statistical Society, Series B*, vol. 39, no. 1, pp. 1–38, 1977.

[22] T. Abatzoglou and B. O. Donnell, "Minimization by coordinate descent," *Journal of Optimization Theory and Applications*, 1982.

[23] W. Zangwill, *Nonlinear Programming: a Unified Approach*. Englewood Cliffs: N.J.: Prentice-Hall International Series in Management, 1969.

[24] C. F. J. Wu, "On the convergence properties of the EM algorithm," *The Annals of Statistics*, vol. 11, no. 1, pp. 95–103, 1983.

[25] I. Csiszar and G. Tusnady, "Information Geometry and Alternating Minimization Procedures," *Statistics and Decisions*, 1984.

[26] A. K. Halberstadt and J. R. Glass, "Heterogeneous acoustic measurements for phonetic classification," in *Proc. Eurospeech '97*, (Rhodes, Greece), pp. 401–404, 1997.

[27] K. F. Lee and H. Hon, "Speaker independant phone recognition using hidden markov models," *IEEE Transactions on Acoustics, Speech and Signal Processing*, vol. 37, no. 11, 1989.

[28] J. Godfrey, E. Holliman, and J. McDaniel, "Switchboard: Telephone speech corpus for research and development," in *Proceedings of ICASSP*, vol. 1, (San Francisco, California), pp. 517–520, March 1992.

[29] N. Deshmukh, A. Ganapathiraju, A. Gleeson, J. Hamaker, and J. Picone, "Resegmentation of switchboard," in *Proceedings of ICSLP*, (Sydney, Australia), pp. 1543–1546, November 1998.

[30] S. Greenberg, "The Switchboard transcription project," tech. rep., The Johns Hopkins University (CLSP) Summer Research Workshop, 1995.

[31] J. Friedman, J. Bentley, and R. Finkel, "An algorithm for finding best matches in logarithmic expected time," *ACM Transaction on Mathematical Software*, vol. 3, 1977.

[32] S. Arya and D. M. Mount, "Approximate nearest neighbor queries in fixed dimensions," in *ACM-SIAM Symp. on Discrete Algorithms (SODA)*, 1993.

